# Optimal Aggregation of Classifiers and Boosting Maps in Functional Magnetic Resonance Imaging

**Vladimir Koltchinskii**
Department of Mathematics and Statistics
University of New Mexico
Albuquerque, NM, 87131

**Manel Martínez-Ramón**
Department of Electrical and Computer Engineering
University of New Mexico
Albuquerque, NM, 87131

**Stefan Posse**
Department of Psychiatry and The Mind Institute
University of New Mexico
Albuquerque, NM, 87131

## Abstract

We study a method of optimal data-driven aggregation of classifiers in a convex combination and establish tight upper bounds on its excess risk with respect to a convex loss function under the assumption that the solution of optimal aggregation problem is sparse. We use a boosting type algorithm of optimal aggregation to develop aggregate classifiers of activation patterns in fMRI based on locally trained SVM classifiers. The aggregation coefficients are then used to design a "boosting map" of the brain needed to identify the regions with most significant impact on classification.

## 1   Introduction

We consider a problem of optimal aggregation (see [1]) of a finite set of base classifiers in a complex aggregate classifier. The aggregate classifiers we study are convex combinations of base classifiers and we are using boosting type algorithms as aggregation tools. Building upon recent developments in learning theory, we show that such boosting type aggregation yields a classifier with a small value of excess risk in the case when optimal aggregate classifiers are sparse and that, moreover, the procedure provides reasonably good estimates of aggregation coefficients. Our primary goal is to use this approach in the problem of classification of activation patterns in functional Magnetic Resonance Imaging (fMRI) (see, e.g., [2]).

In these problems it is of interest not only to classify the patterns, but also to determine areas of the brain that are relevant for a particular classification task. Our approach is based

on splitting the image into a number of functional areas, training base classifiers locally in each area and then combining them into a complex aggregate classifier. The aggregation coefficients are used to create a special representation of the image we call the *boosting map* of the brain. It is needed to identify the functional areas with the most significant impact on classification.

Previous work has focused on classifying patterns within subject [2] and these patterns were located in the occipital lobe. Here we are considering a different problem, that is widely distributed patterns in multiple brain regions across groups of subjects. We use prior knowledge from functional neuroanatomical brain atlases to subdivide the brain into Regions of Interest, which makes this problem amenable to boosting. Classification across subjects requires spatial normalization to account for inter-subject differences in brain size and shape, but also needs to be robust with respect to inter-subject differences in activation patterns –shape and amplitude.

Since fMRI patterns are very high dimensional and the amount of training data is typically limited, some form of "bet on sparsity" principle ("use a procedure that does well in sparse problems, since no procedure does well in dense problems" see [3]) becomes almost unavoidable and our theoretical analysis shows that boosting maps might have a good chance of success in sparse problems (when only few functional areas are relevant for classification).

## 2   Optimal aggregation of classifiers

Although we developed a multiclass extension of the method, for simplicity, we are dealing here with a standard binary classification. Let $(X, Y)$ be a random couple with distribution $P$, $X$ being an instance in some space $S$ (e.g., it might be an fMRI pattern) and $Y \in \{-1, 1\}$ being a binary label. Here and in what follows all the random variables are defined on a probability space $(\Omega, \Sigma, \mathbb{P})$, $\mathbb{E}$ denotes the expectation. Functions $f : S \mapsto \mathbb{R}$ will be used as classifiers, $\text{sign}(f(x))$ being a predictor of the label for an instance $x \in S$ (no decision is being made if $f(x) = 0$). The quantity $P\{(x, y) : yf(x) \leq 0\}$ (the probability of misclassification or abstaining) is called the generalization error or the risk of $f$. Suppose that $\mathcal{H} := \{h_1, \ldots, h_N\}$ is a given family of classifiers taking values in $[-1, 1]$. Let

$$\text{conv}(\mathcal{H}) := \left\{ \sum_{j=1}^{N} \lambda_j h_j : \sum_{j=1}^{N} |\lambda_j| \leq 1 \right\}$$

be the symmetric convex hull of $\mathcal{H}$. One of the versions of *optimal aggregation problem* would be to find a convex combination $f \in \text{conv}(\mathcal{H})$ that minimizes the generalization error of $f$ in $\text{conv}(\mathcal{H})$. For a given $f \in \text{conv}(\mathcal{H})$ its quality is measured by

$$\mathcal{E}(f) := P\{(x, y) : yf(x) \leq 0\} - \inf_{g \in \text{conv}(\mathcal{H})} P\{(x, y) : yg(x) \leq 0\},$$

which is often called the excess risk of $f$. Since the true distribution $P$ of $(X, Y)$ is unknown, the solution of the optimal aggregation problem is to be found based on *the training data* $(X_1, Y_1), \ldots, (X_n, Y_n)$ consisting of $n$ independent copies of $(X, Y)$.

Let $P_n$ denote the empirical measure based on the training data, i.e., $P_n(A)$ represents the frequency of training examples in a set $A \subset S \times \{-1, 1\}$. In what follows, we denote $Ph$ or $P_n h$ the integrals of a function $h$ on $S \times \{-1, 1\}$ with respect to $P$ or $P_n$, respectively. We use the same notation for functions on $S$ with an obvious meaning.

Since the generalization error is not known, it is tempting to try to estimate the optimal convex aggregate classifier by minimizing *the training error* $P_n\{(x, y) : yf(x) \leq 0\}$ over the convex hull $\text{conv}(\mathcal{H})$. However, this minimization problem is not computationally feasible and, moreover, the accuracy of empirical approximation (approximation of $P$ by $P_n$) over the class of sets $\{\{(x, y) : yf(x) \leq 0\} : f \in \text{conv}(\mathcal{H})\}$ is not good enough when $\mathcal{H}$ is a large class. An approach that allows one to overcome both difficulties and that

proved to be very successful in the recent years is to replace the minimization of the training error by the minimization of the empirical risk with respect to a convex loss function. To be specific, let $\ell$ be a nonnegative decreasing convex function on $\mathbb{R}$ such that $\ell(u) \geq 1$ for $u \leq 0$. We will denote $(\ell \bullet f)(x, y) := \ell(yf(x))$. The quantity

$$P(\ell \bullet f) = \int (\ell \bullet f)dP = \mathbb{E}\ell(Yf(X))$$

is called the risk of $f$ with respect to the loss $\ell$, or the $\ell$-risk of $f$. We will call a function

$$f_0 := \sum_{j=1}^{N} \lambda_j^0 h_j \in \text{conv}(\mathcal{H})$$

*an $\ell$-otimal aggregate classifier* if it minimizes the $\ell$-risk over $\text{conv}(\mathcal{H})$. Similarly to the excess risk, one can define the excess $\ell$-risk of $f$ as

$$\mathcal{E}_\ell(f) := P(\ell \bullet f) - \inf_{g \in \text{conv}(\mathcal{H})} P(\ell \bullet g).$$

Despite the fact that we concentrate in what follows on optimizing the excess $\ell$-risk ($\ell$-optimal aggregation) it often provides also a reasonably good solution of the problem of minimizing the generalization error (optimal aggregation), as it follows from simple inequalities relating the two risks and proved in [4].

As before, since $P$ is unknown, the minimization of $\ell$-risk has to be replaced by the corresponding empirical risk minimization problem

$$P_n(\ell \bullet f) = \frac{1}{n} \sum_{i=1}^{n} \ell\Big(Y_j f(X_j)\Big) \longrightarrow \min, \ f \in \text{conv}(\mathcal{H}),$$

whose solution $\hat{f} := \sum_{j=1}^{N} \hat{\lambda}_j h_j$ is called *an empirical $\ell$-optimal aggregate* classifier.

We will show that if $f_0, \hat{f}$ are "sparse" (i.e., $\lambda_j^0, \hat{\lambda}_j$ are small for most of the values of $j$), then the excess $\ell$-risk of the empirical $\ell$-optimal aggregate classifier is small and, moreover, the coefficients of $\hat{f}$ are close to the coefficients of $f_0$ in $\ell_1$-distance.

The sparsity assumption is almost unavoidable in many problems because of the "bet on sparsity" principle (see the Introduction).

At a more formal level, if there exists a small subset $J \subset \{1, 2, \ldots, N\}$ such that the sets of random variables $\{Y, h_j(X), j \in J\}$ and $\{h_j(X), j \notin J\}$ are independent and, in addition, $\mathbb{E}h_j(X) = 0, j \notin J$, then, using Jensen's inequality, it is easy to check that in an $\ell$-optimal aggregate classifier $f_0$ one can take $\lambda_j^0 = 0, j \notin J$.

We will define a measure of sparsity of a function $f := \sum_{j=1}^{N} \lambda_j h_j \in \text{conv}(\mathcal{H})$ that is somewhat akin to sparsity charactersitics considered in [5, 6]. For $0 \leq d \leq N$, let

$$\Delta(f; d) := \min\Big\{\sum_{j \notin J} |\lambda_j| : J \subset \{1, \ldots, N\}, \ \text{card}(J) = d\Big\}$$

and let $\beta_n(d) := \frac{d \log(Nn/d)}{n}$.

Define

$$d_n(f) := \min\Big\{d : 1 \leq d \leq N, \ \sqrt{\beta_n(d)} \geq \Delta(d)\Big\}.$$

Of course, if there exists $J \subset \{1, \ldots, N\}$ such that $\lambda_j = 0$ for all $j \notin J$ and $\text{card}(J) = d$, then $d_n(f) \leq d$.

We will also need the following measure of linear independence of functions in $\mathcal{H}$ :

$$\gamma(d) := \gamma(\mathcal{H}; d) = \left(\inf_{J \subset \{1, \ldots, N\}, \text{card}(J) = d} \ \inf_{\sum_{j \in J} |\alpha_j| = 1} \left\|\sum_{j \in J} \alpha_j h_j\right\|_{L_2(P)}\right)^{-1}.$$

Finally, we need some standard conditions on the loss function $\ell$ (as, for instance, in [4]). Assume that $\ell$ is Lipschitz on $[-1, 1]$ with some constant $L$, $|\ell(u) - \ell(v)| \leq L|u - v|, u, v \in [-1, 1]$, and the following condition on the convexity modulus of $\ell$ holds with $\Lambda \leq L$:

$$\frac{\ell(u) + \ell(v)}{2} - \ell\left(\frac{u + v}{2}\right) \geq \Lambda|u - v|^2, u, v \in [-1, 1].$$

In fact, $\ell(u)$ is often replaced by a function $\ell(uM)$ with a large enough $M$ (in other words, the $\ell$-risk is minimized over $M\mathrm{conv}(\mathcal{H})$). This is the case, for instance, for so called regularized boosting [7]. The theorem below applies to this case as well, only a simple rescaling of the constants is needed.

**Theorem 1** *There exist constants $K_1, K_2 > 0$ such that for all $t > 0$*

$$\mathbb{P}\left\{\mathcal{E}_\ell(\hat{f}) \geq K_1 \frac{L^2}{\Lambda}\left(\beta_n(d_n(\hat{f})) \bigwedge \sqrt{\frac{\log N}{n}} + \frac{t}{n}\right)\right\} \leq e^{-t}$$

*and*

$$\mathbb{P}\left\{\sum_{j=1}^{N} |\hat{\lambda}_j - \lambda_j^0| \geq K_2 \frac{L}{\Lambda}\gamma(d_n(\hat{f}) + d_n(f_0))\sqrt{\beta_n(d_n(\hat{f}) + d_n(f_0)) + \frac{t}{n}}\right\} \leq e^{-t}.$$

Our proof requires some background material on localized Rademacher complexities and their role in bounding of excess risk (see [8]). We defer it to the full version of the paper. Note that the first bound depends only on $d_n(\hat{f})$ and the second on $d_n(\hat{f}), d_n(f_0)$. Both quantities can be much smaller than $N$ despite the fact that empirical risk minimization occurs over the whole $N$-dimensional convex hull. However, the approach to convex aggregation based on minimization of the empirical $\ell$-risk over the convex hull does not guarantee that $\hat{f}$ is sparse even if $f_0$ is. To address this problem, we also studied another approach based on minimization of the *penalized* empirical $\ell$-risk with the penalty based on the number of nonzero coefficients of the classifier, but the size of the paper does not allow us to discuss it.

## 3 Classification of fMRI patterns and boosting maps

We are using optimal aggregation methods described above in the problem of classification of activation patterns in fMRI. Our approach is based on dividing the training data into two parts: for local training and for aggregation. Then, we split the image into $N$ functional areas and train $N$ local classifiers $h_1, \ldots, h_N$ based on the portions of fMRI data corresponding to the areas. The data reserved for aggregation is then used to construct an aggregate classifier. In applications, we are often replacing direct minimization of empirical risk with convex loss by the standard *AdaBoost* algorithm (see, e.g., [9]), which essentially means choosing the loss function as $\ell(u) = e^{-u}$. A weak (base) learner for *AdaBoost* simply chooses in this case a local classifier among $h_1, \ldots, h_N$ with the smallest weighted training error [in more sophisticated versions, we choose a local classifier at random with probability depending on the size of its weighted training error] and after a number of rounds *AdaBoost* returns a convex combination of local classifiers. The coefficients of this aggregate classifier are then used to create a new visual representation of the brain (the boosting map) that highlights the functional areas with significant impact on classification. In principle, it is also possible to use the same data for training of local classifiers and for aggregation (retraining the local classifiers at each round of boosting), but this approach is time consuming.

We use statistical parametric model (SPM) t-maps of MRI scans [10]. Statistical parametric maps (SPMs) are image processes with voxel[1] values that are, under the null hypothesis, distributed according to a known probability density function, usually the Student's

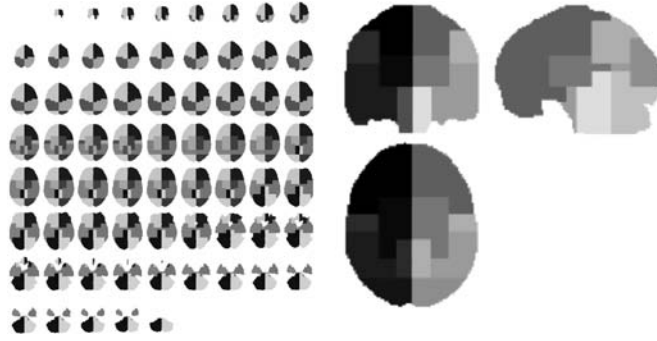

Figure 1: *Masks used to split the image into functional areas in multi-slice and 3 orthogonal slice display representations.*

T or F distributions. These are known colloquially as t- or f-maps. Namely, one analyzes each and every voxel using any standard (univariate) statistical test. The resulting statistical parameters are assembled into an image - the SPM.

The classification system essentially transforms the t-map of the image into the boosting map and at the same time it returns the aggregate classifier. The system consists of **the data preprocessing block** that splits the image into functional areas based on specified masks, and also splits the data into portions corresponding to the areas. In one of our examples, we use the main functional areas *brainstem*, *cerebellum*, *occipital*, *temporal*, *parietal*, *subcortical* and *frontal*. We split these masks in left and right, having in total 14 of them. **The classifier block** then trains local classifiers based on local data (in the current version we are using SVM classifiers). Finally, **the aggregation or boosting block** computes and outputs the aggregate classifier and the boosting map of the image. We developed a version of the system that deals with multi-class problems in spirit of [11], but the details go beyond the scope of this paper. The architecture of the network allows us also to train it sequentially. Let $f$ be a classifier produced by the network in the previous round of work, let $(X_1, Y_1), \ldots, (X_n, Y_n)$ be either the same or a new training data set and let $h_1, \ldots, h_N$ be local classifiers (based either on the same, or on a new set of masks). Then one can assign to the training examples the initial weights $w_j = \frac{e^{-Y_j f(X_j)}}{Z}$, where $Z$ is a standard normalizing constant, instead of usually chosen uniform weights. After this, the *AdaBoost* can proceed in a normal fashion creating at the end an aggregate of $f$ and of new local classifiers. The process can be repeated recursively updating both the classifier and the boosting map.

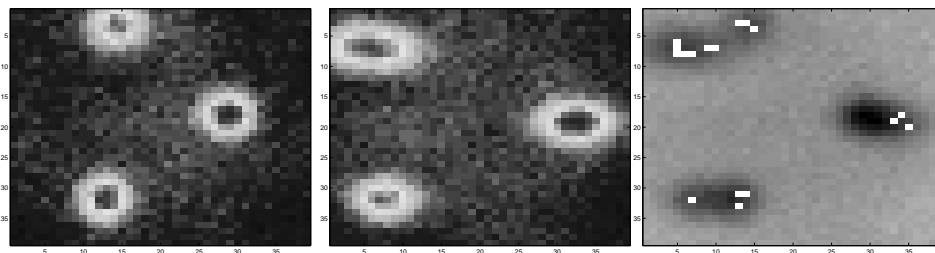

Figure 2: *Left and center: Patterns corresponding to two classes of data. Right: Locations of the learners chosen by the boosting procedure (white spots). The background image corresponds to the two patterns of left and center figures superimposed.*

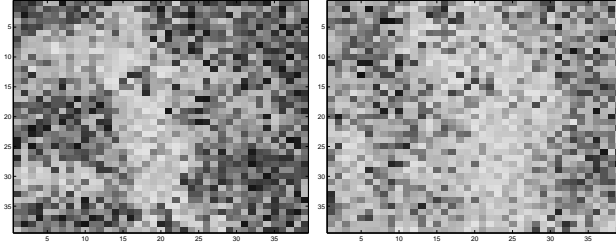

Figure 3: *Patterns corrupted with noise in the gaussian parameters, artifacts, and additive noise used in the synthetic data experiment.*

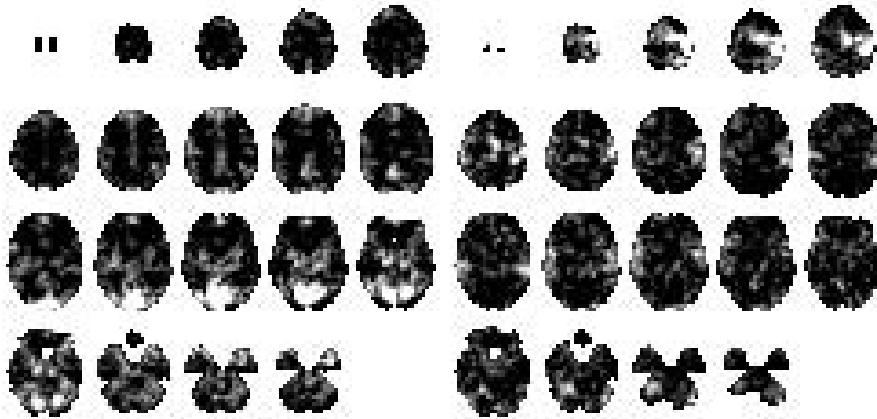

Figure 4: *Two t-maps corresponding to visual (left) and motor activations in the same subject used in the real data experiment.*

As a synthetic data example, we generate $40 \times 40$ pixels images of two classes. Each class of images consists of three gaussian clusters placed in different positions. We generate the set of images by adding gaussian noise of standard deviation 0.1 to the standard deviation and position of the clusters. Then, we add 10 more clusters with random parameters, and finally, additive noise of standard deviation 0.1. Figure 2 (left and center) shows the averages of class 1 and class 2 images respectively. Two samples of the images can be seen in Figure 3

We apply a base learner to each one of the 1600 pixels of the images. Learners have been trained with 200 data, 100 of each class, and the aggregation has been trained with 200 more data. The classifier has been tested with 200 previously unknown data. The error averaged over 100 trials is of $9.5\%$. The same experiment has been made with a single linear SVM, producing an error which exceeds $20\%$, although this rate can be slightly improved by selecting $C$ by cross validation.

The resulting boosting map can be seen in Fig. 2 (right). As a proof of concept, we remark that the map is able to focus in the areas in which the clusters corresponding to each class are, discarding those areas in which only randomly placed clusters are present.

In order to test the algorithm in a real fMRI experiment, we use 20 images taken from 10 healthy subjects on a 1.5 Tesla Siemens Sonata scanner. Stimuli were presented via MR compatible LCD goggles and headphones. The paradigm consists of four interleaved tasks: visual (8 Hz checkerboard stimulation), motor (2 Hz right index finger tapping), auditory

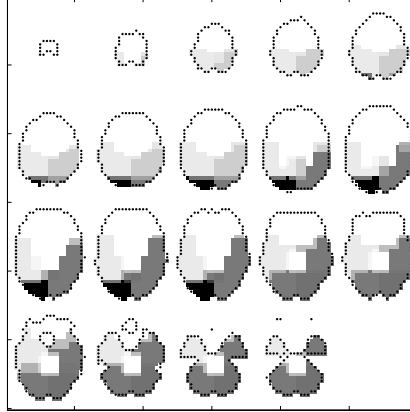

Figure 5: *Boosting map of the brain corresponding to the classification problem with visual and motor activations. Darker regions correspond to higher values.*

| left brainstem: | 0 | right brainstem: | 0 |
|---|---|---|---|
| left cerebellum: | 0.15 | right cerebellum: | 0.16 |
| left parietal: | 0.02 | right parietal: | 0.06 |
| left temporal: | 0.03 | right temporal: | 0.15 |
| left occipital: | 0.29 | right occipital: | 0.15 |
| left subcortical: | 0 | right subcortical: | 0 |
| left frontal: | 0 | right frontal: | 0 |

Table 1: Values of the convex aggregation.

(syllable discrimination) and cognitive (mental calculation). These tasks are arranged in randomized blocks (8 s per block). Finger tapping in the motor task was regulated with an auditory tone, subjects were asked to tap onto a button-response pad. During the auditory task, subjects were asked to respond on a button-response pad for each "Ta" (25% of sounds), but not to similar syllables. Mental calculation stimuli consisted of three single-digit numbers heard via headphone. Participants had to sum them and divide by three, responding by button press when there was no remainder (50% of trials).

Functional MRI data were acquired using single-shot echo-planar imaging with TR: 2 s, TE: 50 ms, flip angle: 90 degrees, matrix size: $64 \times 64$ pixels, FOV: 192 mm. Slices were 6 mm thick, with 25% gap, 66 volumes were collected for a total measurement time of 132 sec per run. Statistical parametric mapping was performed to generate t-maps that represent brain activation changes.

The t-maps are lowpass filtered and undersampled to obtain $32 \times 32 \times 24$ t-maps (Fig. 4). The resulting t-maps are masked to obtain 14 subimages, then the data is normalized in amplitude. We proceed as mentioned to train a set of 14 Support Vector Machines. The used kernel is a gaussian one with $\sigma = 2$ and $C = 10$. These parameters have been chosen to provide an acceptable generalization. A convex aggregation of the classifier outputs is then trained.

We tested the algorithm in binary classification of visual against auditory activations. We train the base learners with 10 images, and the boosting with 9. Then, we train the base learners again with 19, leaving one for testing. We repeat the experiment leaving out a different image each trial. None of the images was misclassified. The values for the aggregation are in Table 1. The corresponding boosting map is shown in Fig 5. It highlights the right temporal and both occipital areas, where the motor and visual activations are

present (see Fig. 4). Also, there is activation in the cerebellum area in some of the motor t-maps, which is highlighted by the boosting map.

In experiments for the six binary combination of activation stimuli, the average error was less than $10\%$. This is an acceptable result if we take into account that the data included ten different subjects, whose brain activation patterns present noticeable differences.

## 4   Future goals

Boosting maps we introduced in this paper might become a useful tool in solving classification problems for fMRI data, but there is a number of questions to be answered before it is the case. The most difficult problem is the choice of functional areas and local classifiers so that the "true" boosting map is identifiable based on the data. As our theoretical analysis shows, this is related to the degree of linear independence of local classifiers quantified by the function $\gamma(d)$. If $\gamma(d)$ is too large for $d = d_n(f_0) \vee d_n(\hat{f})$, the empirical boosting map can become very unstable and misleading. In such cases, there is a challenging model selection problem (how to choose a "good" subset of $\mathcal{H}$ or how to split $\mathcal{H}$ into "almost linearly independent clusters" of functions) that has to be addressed to develop this methodology further.

### Acknowledgments

We want to acknowledge to Jeremy Bockholt (MIND Institute) for providing the brain masks, generated with *BRAINS2*. Partially supported by NSF Grant DMS-0304861 and NIH Grant NIBIB 1 RO1 EB002618-01, Dept. of Mathematics and Statistics, Dept. of Electrical and Computing Engineering, Dept. of Psychiatry and The MIND Institute.

## Footnotes

[1]A voxel is the amplitude of a position in the 3-D MRI image matrix.

## References

[1] Tsybakov, A. (2003) Optimal rates of aggregation. In: COLT2003, *Lecture Notes in Artificial Intelligence,* Eds.: M. Warmuth and B. Schoelkopf, Springer.

[2] Cox, D.D., Savoy, R.L. (2003) Functional magnetic resonance imaging (fMRI) "brain reading": detecting and classifying distributed patterns of fMRI activity in human visual cortex, *Neuroimage*19, 2, 261–70.

[3] Friedman, J., Hastie, T., Rosset, S., Tibshirani, R. and Zhu, J. (2004) Discussion on Boosting, *Annals of Statistics,* 32, 1, 102–107.

[4] Bartlett, P. L., Jordan, M.I., McAuliffe, J. D. (2003) Convexity, classification, and risk bounds. Technical Report 638, Department of Statistics, U.C. Berkeley, 2003. *Journal of the American Statistical Association.*To appear.

[5] Koltchinskii, V., Panchenko, D. and Lozano, F. (2003) Bounding the generalization error of combined classifiers: balancing the dimensionality and the margins. *A. Appl. Prob.* , 13, 1.

[6] Koltchinskii, V., Panchenko, D. and Andonova, S. (2003) Generalization bounds for voting classifiers based on sparsity and clustering. In: COLT2003, *Lecture Notes in Artificial Intelligence*, Eds.: M. Warmuth and B. Schoelkopf, Springer.

[7] Blanchard, G., Lugosi, G. and Vayatis, N. (2003) On the rates of convergence of regularized boosting classifiers. *Journal of Machine Learning Research* 4, 861-894.

[8] Koltchinskii, V. (2003) Local Rademacher Complexities and Oracle Inequalities in Risk Minimization. Preprint.

[9] Schapire, R. E. (1999) A brief Introduction to Boosting. In: Proc. of the 6th Intl. Conf. on Artificial Inteligence.

[10] Friston, K., Frith, C., Liddle, P. and Frackowiak, R. (1991) Comparing functional (PET) images: the assessment of significant change. *J. Cereb. Blood Flow Met.*11, 690-699

[11] Allwein, E. L., Schapire, R. E., and Singer, Y. (2000) Reducing multiclass to binary: A unifying approach for margin classifiers. *J. Machine Learning Research,* 1, 113-141.
